# Adaptive Forward-Backward Greedy Algorithm for Sparse Learning with Linear Models

**Tong Zhang**
Statistics Department
Rutgers University, NJ
tzhang@stat.rutgers.edu

## Abstract

Consider linear prediction models where the target function is a sparse linear combination of a set of basis functions. We are interested in the problem of identifying those basis functions with non-zero coefficients and reconstructing the target function from noisy observations. Two heuristics that are widely used in practice are forward and backward greedy algorithms. First, we show that neither idea is adequate. Second, we propose a novel combination that is based on the forward greedy algorithm but takes backward steps adaptively whenever beneficial. We prove strong theoretical results showing that this procedure is effective in learning sparse representations. Experimental results support our theory.

## 1 Introduction

Consider a set of input vectors $\mathbf{x}_1, \ldots, \mathbf{x}_n \in R^d$, with corresponding desired output variables $y_1, \ldots, y_n$. The task of supervised learning is to estimate the functional relationship $y \approx f(\mathbf{x})$ between the input $\mathbf{x}$ and the output variable $y$ from the training examples $\{(\mathbf{x}_1, y_1), \ldots, (\mathbf{x}_n, y_n)\}$. The quality of prediction is often measured through a loss function $\phi(f(\mathbf{x}), y)$. In this paper, we consider linear prediction model $f(\mathbf{x}) = \mathbf{w}^T \mathbf{x}$. As in boosting or kernel methods, nonlinearity can be introduced by including nonlinear features in $\mathbf{x}$.

We are interested in the scenario that $d \gg n$. That is, there are many more features than the number of samples. In this case, an unconstrained empirical risk minimization is inadequate because the solution overfits the data. The standard remedy for this problem is to impose a constraint on $\mathbf{w}$ to obtain a *regularized* problem. An important target constraint is sparsity, which corresponds to the (non-convex) $L_0$ regularization, where we define $\|\mathbf{w}\|_0 = |\{j : \mathbf{w}_j \neq 0\}| = k$. If we know the sparsity parameter $k$, a good learning method is $L_0$ regularization:

$$\hat{\mathbf{w}} = \arg \min_{\mathbf{w} \in R^d} \frac{1}{n} \sum_{i=1}^n \phi(\mathbf{w}^T \mathbf{x}_i, y_i) \quad \text{subject to } \|\mathbf{w}\|_0 \leq k. \tag{1}$$

If $k$ is not known, then one may regard $k$ as a tuning parameter, which can be selected through cross-validation. This method is often referred to as *subset selection* in the literature. Sparse learning is an essential topic in machine learning, which has attracted considerable interests recently. Generally speaking, one is interested in two closely related themes: feature selection, or identifying the basis functions with non-zero coefficients; estimation accuracy, or reconstructing the target function from noisy observations. It can be shown that the solution of the $L_0$ regularization problem in (1) achieves good prediction accuracy if the target function can be approximated by a sparse $\bar{\mathbf{w}}$. It can also solve the feature selection problem under extra identifiability assumptions. However, a fundamental difficulty with this method is the computational cost, because the number of subsets of $\{1, \ldots, d\}$ of cardinality $k$ (corresponding to the nonzero components of $\mathbf{w}$) is exponential in $k$. There are no efficient algorithms to solve the subset selection formulation (1).

Due to the computational difficult, in practice, there are three standard methods for learning sparse representations by solving approximations of (1). The first approach is $L_1$-regularization (Lasso). The idea is to replace the $L_0$ regularization in (1) by $L_1$ regularization. It is the closest convex approximation to (1). It is known that $L_1$ regularization often leads to sparse solutions. Its performance has been theoretically analyzed recently. For example, if the target is truly sparse, then it was shown in [10] that under some restrictive conditions referred to as *irrepresentable conditions*, $L_1$ regularization solves the feature selection problem. The prediction performance of this method has been considered in [6, 2, 1, 9]. Despite its popularity, there are several problems with $L_1$ regularization: first, the sparsity is not explicitly controlled, and good feature selection property requires strong assumptions; second, in order to obtain very sparse solution, one has to use a large regularization parameter that leads to suboptimal prediction accuracy because the $L_1$ penalty not only shrinks irrelevant features to zero, but also shrinks relevant features to zero. A sub-optimal remedy is to threshold the resulting coefficients; however this requires additional tuning parameters, making the resulting procedures more complex and less robust. The second approach to approximately solve the subset selection problem is *forward greedy algorithm*, which we will describe in details in Section 2. The method has been widely used by practitioners. The third approach is *backward greedy algorithm*. Although this method is widely used by practitioners, there isn't any theoretical analysis when $n \ll d$ (which is the case we are interested in here). The reason will be discussed later.

In this paper, we are particularly interested in greedy algorithms because they have been widely used but the effectiveness has not been well analyzed. As we shall explain later, neither the standard forward greedy idea nor th standard backward greedy idea is adequate for our purpose. However, the flaws of these methods can be fixed by a simple combination of the two ideas. This leads to a novel adaptive forward-backward greedy algorithm which we present in Section 3. The general idea works for all loss functions. For least squares loss, we obtain strong theoretical results showing that the method can solve the feature selection problem under moderate conditions.

For clarity, this paper only considers the fixed design formulation. To simplify notations in our description, we will replace the optimization problem in (1) with a more general formulation. Instead of working with $n$ input data vectors $\mathbf{x}_i \in R^d$, we work with $d$ feature vectors $\mathbf{f}_j \in R^n$ ($j = 1, \ldots, d$), and $\mathbf{y} \in R^n$. Each $\mathbf{f}_j$ corresponds to the $j$-th feature component of $\mathbf{x}_i$ for $i = 1, \ldots, n$. That is, $\mathbf{f}_{j,i} = \mathbf{x}_{i,j}$. Using this notation, we can generally rewrite (1) with in the form $\hat{\mathbf{w}} = \arg\min_{\mathbf{w} \in R^d} R(\mathbf{w})$ subject to $\|\mathbf{w}\|_0 \leq k$, where weight $\mathbf{w} = [\mathbf{w}_1, \ldots, \mathbf{w}_d] \in R^d$, and $R(\mathbf{w})$ is a real-valued cost function which we are interested in optimization. For least squares regression, we have $R(\mathbf{w}) = n^{-1} \|\sum_j \mathbf{w}_j \mathbf{f}_j - \mathbf{y}\|_2^2$. In the following, we also let $\mathbf{e}_j \in R^d$ be the vector of zeros, except for the $j$-component which is one. For convenience, we also introduce the following notations.

**Definition 1.1** *Define* $\text{supp}(\mathbf{w}) = \{j : \mathbf{w}_j \neq 0\}$ *as the set of nonzero coefficients of a vector* $\mathbf{w} = [\mathbf{w}_1, \ldots, \mathbf{w}_d] \in R^d$. *For a weight vector* $\mathbf{w} \in R^d$, *we define mapping* $f : R^d \to R^n$ *as:* $f(\mathbf{w}) = \sum_{j=1}^d \mathbf{w}_j \mathbf{f}_j$. *Given* $\mathbf{f} \in R^d$ *and* $F \subset \{1, \ldots, d\}$, *let* $\hat{\mathbf{w}}(F, \mathbf{f}) = \min_{\mathbf{w} \in R^d} \|f(\mathbf{w}) - \mathbf{f}\|_2^2$ *subject to* $\text{supp}(\mathbf{w}) \subset F$, *and let* $\hat{\mathbf{w}}(F) = \hat{\mathbf{w}}(F, \mathbf{y})$ *be the solution of the least squares problem using features* $F$.

## 2   Forward and Backward Greedy Algorithms

Forward greedy algorithms have been widely used in applications. The basic algorithm is presented in Figure 1. Although a number of variations exist, they all share the basic form of greedily picking an additional feature at every step to aggressively reduce the cost function. The intention is to make most significant progress at each step in order to achieve sparsity. In this regard, the method can be considered as an approximation algorithm for solving (1).

A major flaw of this method is that it can never correct mistakes made in earlier steps. As an illustration, we consider the situation plotted in Figure 2 with least squares regression. In the figure, $\mathbf{y}$ can be expressed as a linear combination of $\mathbf{f}_1$ and $\mathbf{f}_2$ but $\mathbf{f}_3$ is closer to $\mathbf{y}$. Therefore using the forward greedy algorithm, we will find $\mathbf{f}_3$ first, then $\mathbf{f}_1$ and $\mathbf{f}_2$. At this point, we have already found all good features as $\mathbf{y}$ can be expressed by $\mathbf{f}_1$ and $\mathbf{f}_2$, but we are not able to remove $\mathbf{f}_3$ selected in the first step. The above argument implies that forward greedy method is inadequate for feature selection. The method only works when small subsets of the basis functions $\{\mathbf{f}_j\}$ are near orthogonal

Input: $\mathbf{f}_1, \ldots, \mathbf{f}_d, \mathbf{y} \in R^n$ and $\epsilon > 0$
Output: $F^{(k)}$ and $\mathbf{w}^{(k)}$
let $F^{(0)} = \emptyset$ and $\mathbf{w}^{(0)} = 0$
**for** $k = 1, 2, \ldots$
   let $i^{(k)} = \arg\min_i \min_\alpha R(\mathbf{w}^{(k-1)} + \alpha \mathbf{e}_i)$
   let $F^{(k)} = \{i^{(k)}\} \cup F^{(k-1)}$
   let $\mathbf{w}^{(k)} = \hat{\mathbf{w}}(F^{(k)})$
   **if** $(R(\mathbf{w}^{(k-1)}) - R(\mathbf{w}^{(k)}) \leq \epsilon)$ **break**
**end**

Figure 1: Forward Greedy Algorithm

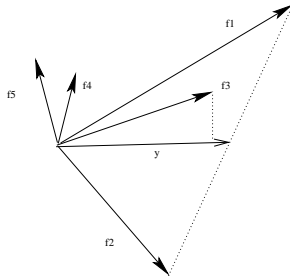

Figure 2: Failure of Forward Greedy Algorithm

[7]. In general, Figure 2 (which is the case we are interested in in this paper) shows that forward greedy algorithm will make errors that are not corrected later on.

In order to remedy the problem, the so-called backward greedy algorithm has been widely used by practitioners. The idea is to train a full model with all the features, and greedily remove one feature (with the smallest increase of cost function) at a time. Although at the first sight, backward greedy method appears to be a reasonable idea that addresses the problem of forward greedy algorithm, it is computationally very costly because it starts with a full model with all features. Moreover, there are no theoretical results showing that this procedure is effective. In fact, under our setting, the method may only work when $d \ll n$ (see, for example, [3]), which is not the case we are interested in. In the case $d \gg n$, during the first step, we start with a model with all features, which can immediately overfit the data with perfect prediction. In this case, the method has no ability to tell which feature is irrelevant and which feature is relevant because removing any feature still completely overfits the data. Therefore the method will completely fail when $d \gg n$, which explains why there is no theoretical result for this method.

## 3   Adaptive Forward-Backward Greedy Algorithm

The main strength of forward greedy algorithm is that it always works with a sparse solution explicitly, and thus computationally efficient. Moreover, it does not significantly overfit the data due to the explicit sparsity. However, a major problem is its inability to correct any error made by the algorithm. On the other hand, backward greedy steps can potentially correct such an error, but need to start with a good model that does not completely overfit the data — it can only correct errors with a small amount of overfitting. Therefore a combination of the two can solve the fundamental flaws of both methods. However, a key design issue is how to implement a backward greedy strategy that is provably effective. Some heuristics exist in the literature, although without any effectiveness proof. For example, the standard heuristics, described in [5] and implemented in SAS, includes another threshold $\epsilon'$ in addition to $\epsilon$: a feature is deleted if the cost-function increase by performing the deletion is no more than $\epsilon'$. Unfortunately we cannot provide an effectiveness proof for this heuristics: if the threshold $\epsilon'$ is too small, then it cannot delete any spurious features introduced in the forward steps; if it is too large, then one cannot make progress because good features are also deleted. In practice it can be hard to pick a good $\epsilon'$, and even the best choice may be ineffective.

This paper takes a more principled approach, where we specifically design a forward-backward greedy procedure with *adaptive* backward steps that are carried out automatically. The procedure has provably good performance and fixes the drawbacks of forward greedy algorithm illustrated in Figure 2. There are two main considerations in our approach: we want to take reasonably aggressive backward steps to remove any errors caused by earlier forward steps, and to avoid maintaining a large number of basis functions; we want to take backward step *adaptively* and make sure that any backward greedy step does not erase the gain made in the forward steps. Our algorithm, which we refer to as *FoBa*, is listed in Figure 3. It is designed to balance the above two aspects. Note that we only take a backward step when the increase of cost function is no more than half of the decrease of cost function in earlier forward steps. This implies that if we take $\ell$ forward steps, then no matter how many backward steps are performed, the cost function is decreased by at least an amount of $\ell\epsilon/2$. It follows that if $R(\mathbf{w}) \geq 0$ for all $\mathbf{w} \in R^d$, then the algorithm terminates after no more than $2R(0)/\epsilon$ steps. This means that the procedure is computationally efficient.

---

Input: $\mathbf{f}_1, \ldots, \mathbf{f}_d, \mathbf{y} \in R^n$ and $\epsilon > 0$
Output: $F^{(k)}$ and $\mathbf{w}^{(k)}$
let $F^{(0)} = \emptyset$ and $\mathbf{w}^{(0)} = 0$
let $k = 0$
**while true**
  let $k = k + 1$
  // forward step
  let $i^{(k)} = \arg\min_i \min_\alpha R(\mathbf{w}^{(k-1)} + \alpha\mathbf{e}_i)$
  let $F^{(k)} = \{i^{(k)}\} \cup F^{(k-1)}$
  let $\mathbf{w}^{(k)} = \hat{\mathbf{w}}(F^{(k)})$
  let $\delta^{(k)} = R(\mathbf{w}^{(k-1)}) - R(\mathbf{w}^{(k)})$
  **if** $(\delta^{(k)} \leq \epsilon)$
    $k = k - 1$
    **break**
  **endif**
  // backward step (can be performed after each few forward steps)
  **while true**
    let $j^{(k)} = \arg\min_{j \in F^{(k)}} R(\mathbf{w}^{(k)} - \mathbf{w}_j^{(k)}\mathbf{e}_j)$
    let $\delta' = R(\mathbf{w}^{(k)} - \mathbf{w}_{j^{(k)}}^{(k)}\mathbf{e}_{j^{(k)}}) - R(\mathbf{w}^{(k)})$
    **if** $(\delta' > 0.5\delta^{(k)})$ **break**
    let $k = k - 1$
    let $F^{(k)} = F^{(k+1)} - \{j^{(k+1)}\}$
    let $\mathbf{w}^{(k)} = \hat{\mathbf{w}}(F^{(k)})$
  **end**
**end**

Figure 3: FoBa: Forward-Backward Greedy Algorithm

---

Now, consider an application of FoBa to the example in Figure 2. Again, in the first three forward steps, we will be able to pick $\mathbf{f}_3$, followed by $\mathbf{f}_1$ and $\mathbf{f}_2$. After the third step, since we are able to express $\mathbf{y}$ using $\mathbf{f}_1$ and $\mathbf{f}_2$ only, by removing $\mathbf{f}_3$ in the backward step, we do not increase the cost. Therefore at this stage, we are able to successfully remove the incorrect basis $\mathbf{f}_3$ while keeping the good features $\mathbf{f}_1$ and $\mathbf{f}_2$. This simple illustration demonstrates the effectiveness of FoBa. In the following, we formally characterize this intuitive example, and prove the effectiveness of FoBa for feature selection as well as parameter estimation. Our analysis assumes the least squares loss. However, it is possible to handle more general loss functions with a more complicated derivation.

We introduce the following definition, which characterizes how linearly independent small subsets of $\{\mathbf{f}_j\}$ of size $k$ are. For $k \ll n$, the number $\rho(k)$ can be bounded away from zero even when $d \gg n$. For example, for random basis functions $\mathbf{f}_j$, we may take $\ln d = O(n/k)$ and still have $\rho(k)$ to be bounded away from zero. This quantity is the smallest eigenvalue of the $k \times k$ diagonal blocks of the $d \times d$ design matrix $[\mathbf{f}_i^T \mathbf{f}_j]_{i,j=1,\ldots,d}$, and has appeared in recent analysis of $L_1$ regularization

methods such as in [2, 8], etc. We shall refer it to as the *sparse eigenvalue condition*. This condition is the least restrictive condition when compared to other conditions in the literature [1].

**Definition 3.1** *Define for all $1 \le k \le d$: $\rho(k) = \inf \left\{ \frac{1}{n} \|f(\mathbf{w})\|_2^2 / \|\mathbf{w}\|_2^2 : \|\mathbf{w}\|_0 \le k \right\}$.*

**Assumption 3.1** *Consider least squares loss $R(\mathbf{w}) = \frac{1}{n} \|f(\mathbf{w}^{(k)}) - \mathbf{y}\|_2^2$. Assume that the basis functions are normalized such that $\frac{1}{n} \|\mathbf{f}_j\|_2^2 = 1$ for all $j = 1, \ldots, d$, and assume that $\{y_i\}_{i=1,\ldots,n}$ are independent (but not necessarily identically distributed) sub-Gaussians: there exists $\sigma \ge 0$ such that $\forall i$ and $\forall t \in R$, $\mathbf{E}_{y_i} e^{t(y_i - \mathbf{E}y_i)} \le e^{\sigma^2 t^2/2}$.*

Both Gaussian and bounded random variables are sub-Gaussian using the above definition. For example, we have the following Hoeffding's inequality. If a random variable $\xi \in [a, b]$, then $\mathbf{E}_\xi e^{t(\xi - \mathbf{E}\xi)} \le e^{(b-a)^2 t^2/8}$. If a random variable is Gaussian: $\xi \sim N(0, \sigma^2)$, then $\mathbf{E}_\xi e^{t\xi} \le e^{\sigma^2 t^2/2}$.

The following theorem is stated with an explicit $\epsilon$ for convenience. In applications, one can always run the algorithm with a smaller $\epsilon$ and use cross-validation to determine the optimal stopping point.

**Theorem 3.1** *Consider the FoBa algorithm in Figure 3, where Assumption 3.1 holds. Assume also that the target is sparse: there exists $\bar{\mathbf{w}} \in R^d$ such that $\bar{\mathbf{w}}^T \mathbf{x}_i = \mathbf{E}y_i$ for $i = 1, \ldots, n$, and $\bar{F} = \mathrm{supp}(\bar{\mathbf{w}})$. Let $\bar{k} = |\bar{F}|$, and $\epsilon > 0$ be the stopping criterion in Figure 3. Let $s \le d$ be an integer which either equals $d$ or satisfies the condition $8\bar{k} \le s\rho(s)^2$. If $\min_{j \in \mathrm{supp}(\bar{\mathbf{w}})} |\bar{\mathbf{w}}_j|^2 \ge \frac{64}{25}\rho(s)^{-2}\epsilon$, and for some $\eta \in (0, 1/3)$, $\epsilon \ge 64\rho(s)^{-2}\sigma^2 \ln(2d/\eta)/n$, then with probability larger than $1 - 3\eta$, when the algorithm terminates, we have $F^{(k)} = \bar{F}$ and $\|\mathbf{w}^{(k)} - \bar{\mathbf{w}}\|_2 \le \sigma\sqrt{\bar{k}/(n\rho(\bar{k}))}\left[1 + \sqrt{20\ln(1/\eta)}\right]$.*

The result shows that one can identify the correct set of features $\bar{F}$ as long as the weights $\bar{\mathbf{w}}_j$ are not close to zero when $j \in \bar{F}$. This condition is necessary for all feature selection algorithms including previous analysis of Lasso. The theorem can be applied as long as eigenvalues of small $s \times s$ diagonal blocks of the design matrix $[\mathbf{f}_i^T \mathbf{f}_j]_{i,j=1,\ldots,d}$ are bounded away from zero (i.e., sparse eigenvalue condition). This is the situation under which the forward greedy step can make mistakes, but such mistakes can be corrected using FoBa. Because the conditions of the theorem do not prevent forward steps from making errors, the example described in Figure 2 indicates that it is not possible to prove a similar result for the forward greedy algorithm. The result we proved is also better than that of Lasso, which can successfully select features under *irrepresentable conditions* of [10]. It is known that the sparse eigenvalue condition considered here is generally weaker [8, 1].

Our result relies on the assumption that $|\bar{\mathbf{w}}_j|$ ($j \in \bar{F}$) is larger than the noise level $O(\sigma\sqrt{\ln d/n})$ in order to select features effectively. If any nonzero weight is below the noise level, then no algorithm can distinguish it from zero with large probability. That is, in this case, one cannot reliably perform feature selection due to the noise. Therefore FoBa is near optimal in term of its ability to perform reliable feature selection, except for the constant hiding in $O(\cdot)$. For target that is not truly sparse, similar results can be obtained. In this case, it is not possible to correctly identify all the features with large probability. However, we can show that FoBa can still select part of the features reliably, with good parameter estimation accuracy. Such results can be found in the full version of the paper, available from the author's website.

# 4 Experiments

We compare FoBa described in Section 3) to forward-greedy and $L_1$-regularization on artificial and real data. They show that in practice, FoBa is closer to subset selection than the other two approaches, in the sense that FoBa achieves smaller training error given any sparsity level. In oder to compare with Lasso, we use the LARS [4] package in R, which generates a path of actions for adding and deleting features, along the $L_1$ solution path. For example, a path of $\{1, 3, 5, -3, \ldots\}$ means that in the fist three steps, feature $1, 3, 5$ are added; and the next step removes feature 3. Using such a solution path, we can compare Lasso to Forward-greedy and FoBa under the same framework. Similar to the Lasso path, FoBa also generates a path with both addition and deletion operations, while forward-greedy algorithm only adds features without deletion.

Our experiments compare the performance of the three algorithms using the corresponding feature addition/deletion paths. We are interested in features selected by the three algorithms at any sparsity level $k$, where $k$ is the desired number of features presented in the final solution. Given a path, we can keep an active feature set by adding or deleting features along the path. For example, for path $\{1, 3, 5, -3\}$, we have two potential active feature sets of size $k = 2$: $\{1, 3\}$ (after two steps) and $\{1, 5\}$ (after four steps). We then define the $k$ best features as the active feature set of size $k$ with the smallest least squares error because this is the best approximation to subset selection (along the path generated by the algorithm). From the above discussion, we do not have to set $\epsilon$ explicitly in the FoBa procedure. Instead, we just generate a solution path which is five times as long as the maximum desired sparsity $k$, and then generate the best $k$ features for any sparsity level using the above described procedure.

## 4.1   Simulation Data

Since for real data, we do not know the true feature set $\bar{F}$, simulation is needed to compare feature selection performance. We generate $n = 100$ data points of dimension $d = 500$. The target vector $\bar{\mathbf{w}}$ is truly sparse with $\bar{k} = 5$ nonzero coefficients generated uniformly from 0 to 10. The noise level is $\sigma^2 = 0.1$. The basis functions $\mathbf{f}_j$ are randomly generated with moderate correlation: that is, some basis functions are correlated to the basis functions spanning the true target. Note that if there is no correlation (i.e., $\mathbf{f}_j$ are independent random vectors), then both forward-greedy and $L_1$-regularization work well because the basis functions are near orthogonal (this is the well-known case considered in the compressed sensing literature). Therefore in this experiment, we generate moderate correlation so that the performance of the three methods can be differentiated. Such moderate correlation does not violate the sparse eigenvalue condition in our analysis, but violates the more restrictive conditions for forward-greedy method and Lasso.

|  | FoBa | Forward-greedy | $L_1$ |
|---|---|---|---|
| least squares training error | $0.093 \pm 0.02$ | $0.16 \pm 0.089$ | $0.25 \pm 0.14$ |
| parameter estimation error | $0.057 \pm 0.2$ | $0.52 \pm 0.82$ | $1.1 \pm 1$ |
| feature selection error | $0.76 \pm 0.98$ | $1.8 \pm 1.1$ | $3.2 \pm 0.77$ |

Table 1: Performance comparison on simulation data at sparsity level $k = 5$

Table 1 shows the performance of the three methods (including two versions of FoBa), where we repeat the experiments 50 times, and report the average ± standard-deviation. We use the three methods to select five best features, using the procedure described above. We report three metrics. Training error is the squared error of the least squares solution with the selected five features. Parameter estimation error is the 2-norm of the estimated parameter (with the five features) minus the true parameter. Feature selection error is the number of incorrectly selected features. It is clear from the table that for this data, FoBa achieves significantly smaller training error than the other two methods, which implies that it is closest to subset selection. Moreover, the parameter estimation performance and feature selection performance are also better. The two versions of FoBa perform very similarly for this data.

## 4.2   Real Data

Instead of listing results for many datasets without gaining much insights, we present a more detailed study on a typical dataset, which reflect typical behaviors of the algorithms. Our study shows that FoBa does what it is designed to do well: that is, it gives a better approximation to subset selection than either forward-greedy or $L_1$ regularization. Moreover, the difference between aggressive FoBa and conservative FoBa become more significant.

In this study, we use the standard *Boston Housing* data, which is the housing data for 506 census tracts of Boston from the 1970 census, available from the *UCI Machine Learning Database Repository*: *http://archive.ics.uci.edu/ml/*. Each census tract is a data-point, with 13 features (we add a constant offset one as the 14th feature), and the desired output is the housing price. In the experiment, we randomly partition the data into 50 training plus 456 test points. We perform the experiments 50 times, and for each sparsity level from 1 to 10, we report the average training and test squared error. The results are plotted in Figure 4. From the results, we can see that FoBa achieves

better training error for any given sparsity, which is consistent with the theory and the design goal of FoBa. Moreover, it achieves better test accuracy with small sparsity level (corresponding to a more sparse solution). With large sparsity level (corresponding to a less sparse solution), the test error increase more quickly with FoBa. This is because it searches a larger space by more aggressively mimic subset selection, which makes it more prone to overfitting. However, at the best sparsity level of 2 or 3 (for aggressive and conservative FoBa, respectively), FoBa achieves significantly better test error. Moreover, we can observe with small sparsity level (a more sparse solution), $L_1$ regularization performs poorly, due to the bias caused by using a large $L_1$-penalty.

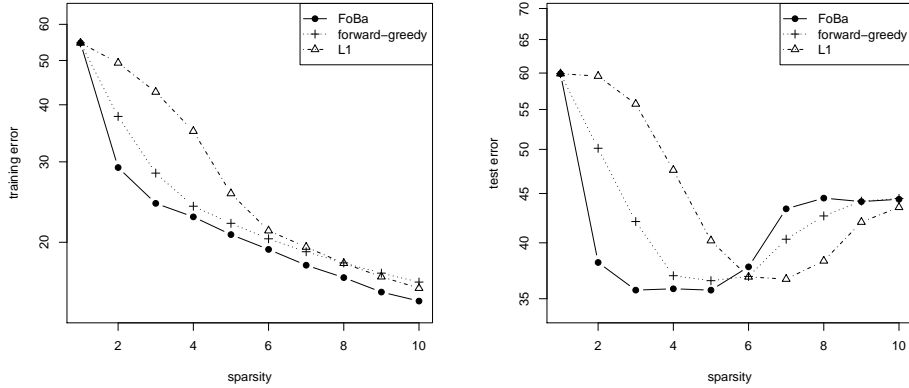

Figure 4: Performance of the algorithms on *Boston Housing* data Left: average training squared error versus sparsity; Right: average test squared error versus sparsity

For completeness, we also compare FoBa to the backward-greedy algorithm and the classical heuristic forward-backward greedy algorithm as implemented in SAS (see its description at the beginning of Section 3). We still use the Boston Housing data, but plot the results separately, in order to avoid cluttering. As we have pointed out, there is no theory for the SAS version of forward-backward greedy algorithm. It is difficult to select an appropriate backward threshold $\epsilon'$: a too small value leads to few backward steps, and a too large value leads to overly aggressive deletion, and the procedure terminates very early. In this experiment, we pick a value of 10, because it is a reasonably large quantity that does not lead to an extremely quick termination of the procedure. The performance of the algorithms are reported in Figure 5. From the results, we can see that backward greedy algorithm performs reasonably well on this problem. Note that for this data, $d \ll n$, which is the scenario that backward does not start with a completely overfitted full model. Still, it is inferior to FoBa at small sparsity level, which means that some degree of overfitting still occurs. Note that backward-greedy algorithm cannot be applied in our simulation data experiment, because $d \gg n$ which causes immediate overfitting. From the graph, we also see that FoBa is more effective than the SAS implementation of forward-backward greedy algorithm. The latter does not perform significant better than the forward-greedy algorithm with our choice of $\epsilon'$. Unfortunately, using a larger backward threshold $\epsilon'$ will lead to an undesirable early termination of the algorithm. This is why the provably effective adaptive backward strategies introduced in this paper are superior.

## 5 Discussion

This paper investigates the problem of learning sparse representations using greedy algorithms. We showed that neither forward greedy nor backward greedy algorithms are adequate by themselves. However, through a novel combination of the two ideas, we showed that an adaptive forward-back greedy algorithm, referred to as FoBa, can effectively solve the problem under reasonable conditions. FoBa is designed to be a better approximation to subset selection. Under the sparse eigenvalue condition, we obtained strong performance bounds for FoBa for feature selection and parameter estimation. In fact, to the author's knowledge, in terms of sparsity, the bounds developed for FoBa in this paper are superior to all earlier results in the literature for other methods.

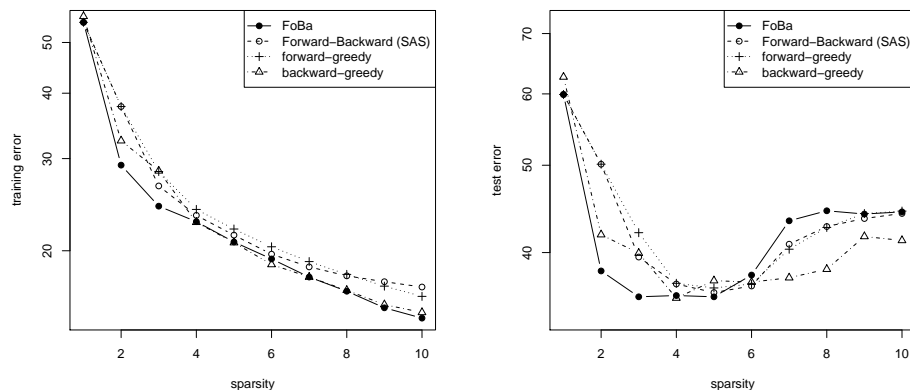

Figure 5: Performance of greedy algorithms on *Boston Housing* data. Left: average training squared error versus sparsity; Right: average test squared error versus sparsity

Our experiments also showed that FoBa achieves its design goal: that is, it gives smaller training error than either forward-greedy or $L_1$ regularization for any given level of sparsity. Therefore the experiments are consistent with our theory. In real data, better sparsity helps on some data such as *Boston Housing*. However, we shall point out that while FoBa always achieves better training error for a given sparsity in our experiments on other datasets (thus it achieves our design goal), $L_1$-regularization some times achieves better test performance. This is not surprising because sparsity is not always the best complexity measure for all problems. In particular, the prior knowledge of using small weights, which is encoded in the $L_1$ regularization formulation but not in greedy algorithms, can lead to better generalization performance on some data (when such a prior is appropriate).

## References

[1] Peter Bickel, Yaacov Ritov, and Alexandre Tsybakov. Simultaneous analysis of Lasso and Dantzig selector. *Annals of Statistics*, 2008. to appear.

[2] Florentina Bunea, Alexandre Tsybakov, and Marten H. Wegkamp. Sparsity oracle inequalities for the Lasso. *Electronic Journal of Statistics*, 1:169–194, 2007.

[3] Christophe Couvreur and Yoram Bresler. On the optimality of the backward greedy algorithm for the subset selection problem. *SIAM J. Matrix Anal. Appl.*, 21(3):797–808, 2000.

[4] Bradley Efron, Trevor Hastie, Iain Johnstone, and Robert Tibshirani. Least angle regression. *Annals of Statistics*, 32(2):407–499, 2004.

[5] T. Hastie, R. Tibshirani, and J. Friedman. *The Elements of Statistical Learning*. Springer, 2001.

[6] Vladimir Koltchinskii. Sparsity in penalized empirical risk minimization. *Annales de l'Institut Henri Poincaré*, 2008.

[7] Joel A. Tropp. Greed is good: Algorithmic results for sparse approximation. *IEEE Trans. Info. Theory*, 50(10):2231–2242, 2004.

[8] Cun-Hui Zhang and Jian Huang. Model-selection consistency of the Lasso in high-dimensional linear regression. Technical report, Rutgers University, 2006.

[9] Tong Zhang. Some sharp performance bounds for least squares regression with $L_1$ regularization. *The Annals of Statistics*, 2009. to appear.

[10] Peng Zhao and Bin Yu. On model selection consistency of Lasso. *Journal of Machine Learning Research*, 7:2541–2567, 2006.

